# A New Discriminative Kernel From Probabilistic Models

K. Tsuda,*† M. Kawanabe,* G. Rätsch,§*S. Sonnenburg,* and K.-R. Müller*‡

† AIST CBRC, 2-41-6, Aomi, Koto-ku, Tokyo, 135-0064, Japan
*Fraunhofer FIRST, Kekuléstr. 7, 12489 Berlin, Germany
§Australian National University,
Research School for Information Sciences and Engineering,
Canberra, ACT 0200, Australia
‡University of Potsdam, Am Neuen Palais 10, 14469 Potsdam, Germany
koji.tsuda@aist.go.jp, nabe@first.fraunhofer.de,
Gunnar.Raetsch@anu.edu.au, {sonne, klaus}@first.fraunhofer.de

## Abstract

Recently, Jaakkola and Haussler proposed a method for constructing kernel functions from probabilistic models. Their so called "Fisher kernel" has been combined with discriminative classifiers such as SVM and applied successfully in e.g. DNA and protein analysis. Whereas the Fisher kernel (FK) is calculated from the marginal log-likelihood, we propose the TOP kernel derived from Tangent vectors Of Posterior log-odds. Furthermore we develop a theoretical framework on feature extractors from probabilistic models and use it for analyzing FK and TOP. In experiments our new discriminative TOP kernel compares favorably to the Fisher kernel.

## 1 Introduction

In classification tasks, learning enables us to predict the output $y \in \{-1, +1\}$ of some unknown system given the input $\boldsymbol{x} \in \mathcal{X}$ based on the training examples $\{\boldsymbol{x}_i, y_i\}_{i=1}^n$. The purpose of a feature extractor $\boldsymbol{f} : \mathcal{X} \to \mathbb{R}^D$ is to convert the representation of data without losing the information needed for classification [3]. When $\mathcal{X}$ is a vector space like $\mathbb{R}^d$, a variety of feature extractors have been proposed (e.g. Chapter 10 in [3]). However, they are typically not applicable when $\mathcal{X}$ is a set of sequences of symbols and does not have the structure of a vector space as in DNA or protein analysis [2].

Recently, the Fisher kernel (FK) [6] was proposed to compute features from a probabilistic model $p(\boldsymbol{x}, y|\boldsymbol{\theta})$. At first, the parameter estimate $\hat{\boldsymbol{\theta}}$ is obtained from training examples. Then, the tangent vector of the log marginal likelihood $\log p(\boldsymbol{x}|\hat{\boldsymbol{\theta}})$ is used as a feature vector. The Fisher kernel refers to the inner product in this feature space, but the method is effectively a feature extractor (also since the features are computed explicitly). The Fisher kernel was combined with a discriminative classifier such as SVM and achieved excellent classification results in several fields, for example in DNA and protein analysis [6, 5]. Empirically, it is reported that the FK-SVM system often outperforms the classification performance of the plug-in es-

timate.[1] Note that the Fisher kernel is only one possible member in the family of feature extractors $\boldsymbol{f}_{\hat{\boldsymbol{\theta}}}(\boldsymbol{x}) : \mathcal{X} \to \mathbb{R}^D$ that can be derived from probabilistic models. We call this family "model-dependent feature extractors". Exploring this family is a very important and interesting subject.

Since model-dependent feature extractors have been newly developed, the performance measures for them are not yet established. We therefore first propose two performance measures. Then, we define a new kernel (or equivalently a feature extractor) derived from the Tangent vector Of Posterior log-odds – that we denote as TOP kernel. We will analyze the performance of the TOP kernel and the Fisher kernel in terms of our performance measures. Then the TOP kernel is compared favorably to the Fisher kernel in a protein classification experiment.

## 2    Performance Measures

To begin with, let us describe the notations. Let $\boldsymbol{x} \in \mathcal{X}$ be the input 'point' and $y \in \{-1, +1\}$ be the class label. $\mathcal{X}$ may be a finite set or an infinite set like $\mathbb{R}^d$. Let us assume that we know the parametric model of the joint probability $p(\boldsymbol{x}, y|\boldsymbol{\theta})$ where $\boldsymbol{\theta} \in \mathbb{R}^p$ is the parameter vector. Assume that the model $p(\boldsymbol{x}, y|\boldsymbol{\theta})$ is regular [7] and contains the true distribution. Then the true parameter $\boldsymbol{\theta}^*$ is uniquely determined. Let $\hat{\boldsymbol{\theta}}$ be a consistent estimator [1] of $\boldsymbol{\theta}^*$, which is obtained by $n$ training examples drawn i.i.d. from $p(\boldsymbol{x}, y|\boldsymbol{\theta}^*)$. Let $\partial_{\theta_i} f = \partial f/\partial \theta_i$, $\nabla_{\boldsymbol{\theta}} f = (\partial_{\theta_1} f, \dots, \partial_{\theta_p} f)^\top$, and $\nabla_{\boldsymbol{\theta}}^2 f$ denote a $p \times p$ matrix whose $(i, j)$ element is $\partial^2 f/(\partial \theta_i \partial \theta_j)$.

As the Fisher kernel is commonly used in combination with linear classifiers such as SVMs, one reasonable performance measure is the classification error of a linear classifier $\boldsymbol{w}^T \boldsymbol{f}_{\hat{\boldsymbol{\theta}}}(\boldsymbol{x}) + b$ ($\boldsymbol{w} \in \mathbb{R}^D$ and $b \in \mathbb{R}$) in the feature space. Usually $\boldsymbol{w}$ and $b$ are learned, so the optimal feature extractor is different with regard to each learning algorithm. To cancel out this ambiguity and to make a theoretical analysis possible, we assume the optimal learning algorithm is used. When $\boldsymbol{w}$ and $b$ are optimally chosen, the classification error is

$$R(\boldsymbol{f}_{\hat{\boldsymbol{\theta}}}) = \min_{\boldsymbol{w} \in \mathcal{S}, b \in \mathbb{R}} E_{\boldsymbol{x},y} \Phi[-y(\boldsymbol{w}^\top \boldsymbol{f}_{\hat{\boldsymbol{\theta}}}(\boldsymbol{x}) + b)], \qquad (2.1)$$

where $\mathcal{S} = \{\boldsymbol{w}| \|\boldsymbol{w}\| = 1, \boldsymbol{w} \in \mathbb{R}^D\}$, $\Phi[a]$ is the step function which is 1 when $a > 0$ and otherwise 0, and $E_{\boldsymbol{x},y}$ denotes the expectation with respect to the true distribution $p(\boldsymbol{x}, y|\boldsymbol{\theta}^*)$. $R(\boldsymbol{f}_{\hat{\boldsymbol{\theta}}})$ is at least as large as the Bayes error $L^*$ [3] and $R(\boldsymbol{f}_{\hat{\boldsymbol{\theta}}}) = L^*$ only if the linear classifier implements the same decision rule as the Bayes optimal rule.

As a related measure, we consider the estimation error of the posterior probability by a logistic regressor $F(\boldsymbol{w}^\top \boldsymbol{f}_{\hat{\boldsymbol{\theta}}}(\boldsymbol{x}) + b)$, with e.g. $F(t) = 1/(1 + \exp(-t))$:

$$D(\boldsymbol{f}_{\hat{\boldsymbol{\theta}}}) = \min_{\boldsymbol{w} \in \mathbb{R}^D, b \in \mathbb{R}} E_{\boldsymbol{x}} |F(\boldsymbol{w}^\top \boldsymbol{f}_{\hat{\boldsymbol{\theta}}}(\boldsymbol{x}) + b) - P(y = +1|\boldsymbol{x}, \boldsymbol{\theta}^*)|. \qquad (2.2)$$

The relationship between $D(\boldsymbol{f}_{\hat{\boldsymbol{\theta}}})$ and $R(\boldsymbol{f}_{\hat{\boldsymbol{\theta}}})$ is illustrated as follows: Let $\hat{L}$ be the classification error rate of a posterior probability estimator $\hat{P}(y = +1|\boldsymbol{x})$. With regard to $\hat{L}$, the following inequality is known[1]:

$$\hat{L} - L^* \leq 2E_{\boldsymbol{x}} |\hat{P}(y = +1|\boldsymbol{x}) - P(y = +1|\boldsymbol{x}, \boldsymbol{\theta}^*)|. \qquad (2.3)$$

When $\hat{P}(y = +1|\boldsymbol{x}) := F(\boldsymbol{w}^\top \boldsymbol{f}_{\hat{\boldsymbol{\theta}}}(\boldsymbol{x}) + b)$, this inequality leads to the following relationship between the two measures

$$R(\boldsymbol{f}_{\hat{\boldsymbol{\theta}}}) - L^* \leq 2D(\boldsymbol{f}_{\hat{\boldsymbol{\theta}}}). \qquad (2.4)$$

Since $D(\boldsymbol{f}_{\boldsymbol{\theta}})$ is an upper bound on $R(\boldsymbol{f}_{\boldsymbol{\theta}})$, it is useful to derive a new kernel to minimize $D(\boldsymbol{f}_{\boldsymbol{\theta}})$, as will be done in Sec. 4.

## 3 The Fisher kernel

The Fisher kernel (FK) is defined[2] as $K(\boldsymbol{x}, \boldsymbol{x}') = \boldsymbol{s}(\boldsymbol{x}, \hat{\boldsymbol{\theta}})^{\top} Z^{-1}(\hat{\boldsymbol{\theta}}) \boldsymbol{s}(\boldsymbol{x}', \hat{\boldsymbol{\theta}})$, where $\boldsymbol{s}$ is the Fisher score

$$\boldsymbol{s}(\boldsymbol{x}, \hat{\boldsymbol{\theta}}) = \left( \partial_{\theta_1} \log p(\boldsymbol{x}|\hat{\boldsymbol{\theta}}), \dots, \partial_{\theta_p} \log p(\boldsymbol{x}|\hat{\boldsymbol{\theta}}) \right)^{\top} = \nabla_{\boldsymbol{\theta}} \log p(\boldsymbol{x}, \hat{\boldsymbol{\theta}}),$$

and $Z$ is the Fisher information matrix: $Z(\boldsymbol{\theta}) = \mathrm{E}_{\boldsymbol{x}} \left[ \boldsymbol{s}(\boldsymbol{x}, \boldsymbol{\theta}) \boldsymbol{s}(\boldsymbol{x}, \boldsymbol{\theta})^{\top} \middle| \boldsymbol{\theta} \right]$. The theoretical foundation of FK is described in the following theorem [6]: "a kernel classifier employed the Fisher kernel derived from a model that contains the label as a latent variable is, asymptotically, at least as good a classifier as the MAP labeling based on the model". The theorem says that the Fisher kernel can perform at least as well as the plug-in estimate, if the parameters of linear classifier are properly determined (cf. Appendix A of [6]). With our performance measure, this theorem can be represented more concisely: $R(\boldsymbol{f}_{\boldsymbol{\theta}})$ is bounded by the classification error of the plug-in estimate

$$R(\boldsymbol{f}_{\hat{\boldsymbol{\theta}}}) \leq E_{\boldsymbol{x},y} \Phi[-y(P(y = +1|\boldsymbol{x}, \hat{\boldsymbol{\theta}}) - 0.5)]. \tag{3.1}$$

Note that the classification rule constructed by the plug-in estimate $P(y = +1|\boldsymbol{x}, \hat{\boldsymbol{\theta}})$ can also be realized by a linear classifier in feature space. Property (3.1) is important since it guarantees that the Fisher kernel performs better when the optimal $\boldsymbol{w}$ and $b$ are available. However, the Fisher kernel is not the only one to satisfy this inequality. In the next section, we present a new kernel which satisfies (3.1) and has a more appealing theoretical property as well.

## 4 The TOP Kernel

**Definition** Now we proceed to propose a new kernel: Our aim is to obtain a feature extractor that achieves small $D(\boldsymbol{f}_{\hat{\boldsymbol{\theta}}})$. When a feature extractor $\boldsymbol{f}_{\boldsymbol{\theta}}(\boldsymbol{x})$ satisfies[3]

$$\boldsymbol{w}^{\top} \boldsymbol{f}_{\boldsymbol{\theta}}(\boldsymbol{x}) + b = F^{-1}(P(y = +1|\boldsymbol{x}, \boldsymbol{\theta}^*)) \text{ for all } \boldsymbol{x} \in \mathcal{X} \tag{4.1}$$

with certain values of $\boldsymbol{w}$ and $b$, we have $D(\boldsymbol{f}_{\boldsymbol{\theta}}) = 0$. However, since the true parameter $\boldsymbol{\theta}^*$ is unknown, all we can do is to construct $\boldsymbol{f}_{\boldsymbol{\theta}}$ which approximately satisfies (4.1). Let us define

$$v(\boldsymbol{x}, \boldsymbol{\theta}) = F^{-1}(P(y = +1|\boldsymbol{x}, \boldsymbol{\theta})) = \log(P(y = +1|\boldsymbol{x}, \boldsymbol{\theta})) - \log(P(y = -1|\boldsymbol{x}, \boldsymbol{\theta})),$$

which is called the posterior log-odds of a probabilistic model [1]. By Taylor expansion around the estimate $\hat{\boldsymbol{\theta}}$ up to the first order[4], we can approximate $v(\boldsymbol{x}, \boldsymbol{\theta}^*)$ as

$$v(\boldsymbol{x}, \boldsymbol{\theta}^*) \approx v(\boldsymbol{x}, \hat{\boldsymbol{\theta}}) + \sum_{i=1}^{p} \partial_{\theta_i} v(\boldsymbol{x}, \hat{\boldsymbol{\theta}})(\theta_i^* - \hat{\theta}_i). \tag{4.2}$$

Thus, by setting

$$\boldsymbol{f}_{\hat{\boldsymbol{\theta}}}(\boldsymbol{x}) := (v(\boldsymbol{x}, \hat{\boldsymbol{\theta}}), \partial_{\theta_1} v(\boldsymbol{x}, \hat{\boldsymbol{\theta}}), \dots, \partial_{\theta_p} v(\boldsymbol{x}, \hat{\boldsymbol{\theta}}))^{\top} \qquad (4.3)$$

and

$$\boldsymbol{w} := \boldsymbol{w}^* = (1, \theta_1^* - \hat{\theta}_1, \cdots, \theta_p^* - \hat{\theta}_p)^{\top}, \ b = 0, \qquad (4.4)$$

equation (4.1) is approximately satisfied. Since a Tangent vector Of the Posterior log-odds constitutes the main part of the feature vector, we call the inner product of the two feature vectors "TOP kernel":

$$K(\boldsymbol{x}, \boldsymbol{x}') = \boldsymbol{f}_{\hat{\boldsymbol{\theta}}}(\boldsymbol{x})^{\top} \boldsymbol{f}_{\hat{\boldsymbol{\theta}}}(\boldsymbol{x}'). \qquad (4.5)$$

It is easy to verify that the TOP kernel satisfies (3.1), because we can construct the same decision rule as the plug-in estimate by using the first element only (i.e. $\boldsymbol{w} = (1, 0, \dots, 0)$, $b = 0$).

A Theoretical Analysis In this section, we compare the TOP kernel with the plug-in estimate in terms of performance measures. Later on, we assume that $0 < P(y = +1|\boldsymbol{x}, \boldsymbol{\theta}) < 1$ to prevent $|v(\boldsymbol{x}, \boldsymbol{\theta})|$ from going to infinity. Also, it is assumed that $\nabla_{\boldsymbol{\theta}} P(y = +1|\boldsymbol{x}, \boldsymbol{\theta})$ and $\nabla_{\boldsymbol{\theta}}^2 P(y = +1|\boldsymbol{x}, \boldsymbol{\theta})$ are bounded. Substituting the plug-in estimate denoted by the subscript $\pi$ into $D(\boldsymbol{f}_{\hat{\boldsymbol{\theta}}})$, we have

$$D_{\pi}(\hat{\boldsymbol{\theta}}) = E_{\boldsymbol{x}}|P(y = +1|\boldsymbol{x}, \hat{\boldsymbol{\theta}}) - P(y = +1|\boldsymbol{x}, \boldsymbol{\theta}^*)|.$$

Define $\Delta\boldsymbol{\theta} = \hat{\boldsymbol{\theta}} - \boldsymbol{\theta}^*$. By Taylor expansion around $\boldsymbol{\theta}^*$, we have

$$
\begin{aligned}
D_{\pi}(\hat{\boldsymbol{\theta}}) &= E_x|(\Delta\boldsymbol{\theta})^{\top}\nabla_{\boldsymbol{\theta}} P(y = +1|\boldsymbol{x}, \boldsymbol{\theta}^*) + \frac{1}{2}(\Delta\boldsymbol{\theta})^{\top}\nabla_{\boldsymbol{\theta}}^2 P(y = +1|\boldsymbol{x}, \boldsymbol{\theta}_0)(\Delta\boldsymbol{\theta})| \\
&= O(\|\Delta\boldsymbol{\theta}\|). \qquad (4.6)
\end{aligned}
$$

where $\boldsymbol{\theta}_0 = \boldsymbol{\theta}^* + \gamma\Delta\boldsymbol{\theta}$ $(0 \le \gamma \le 1)$. When the TOP kernel is used,

$$D(\boldsymbol{f}_{\hat{\boldsymbol{\theta}}}) \le E_{\boldsymbol{x}}|F((\boldsymbol{w}^*)^{\top}\boldsymbol{f}_{\hat{\boldsymbol{\theta}}}(\boldsymbol{x})) - P(y = +1|\boldsymbol{x}, \boldsymbol{\theta}^*)|, \qquad (4.7)$$

where $\boldsymbol{w}^*$ is defined as in (4.4). Since $F$ is Lipschitz-continuous, there is a finite positive constant $M$ such that $|F(a) - F(b)| \le M|a - b|$. Thus,

$$D(\boldsymbol{f}_{\hat{\boldsymbol{\theta}}}) \le M E_{\boldsymbol{x}}|(\boldsymbol{w}^*)^{\top}\boldsymbol{f}_{\hat{\boldsymbol{\theta}}}(\boldsymbol{x}) - F^{-1}(P(y = +1|\boldsymbol{x}, \boldsymbol{\theta}^*))|. \qquad (4.8)$$

Since $(\boldsymbol{w}^*)^{\top}\boldsymbol{f}_{\hat{\boldsymbol{\theta}}}(\boldsymbol{x})$ is the Taylor expansion of $F^{-1}(P(y = +1|\boldsymbol{x}, \boldsymbol{\theta}^*))$ up to the first order (4.2), the first order terms of $\Delta\boldsymbol{\theta}$ are excluded from the right side of (4.8), thus $D(\boldsymbol{f}_{\hat{\boldsymbol{\theta}}}) = O(\|\Delta\boldsymbol{\theta}\|^2)$. Since both, the plug-in and the TOP kernel, depend on the parameter estimate $\hat{\boldsymbol{\theta}}$, the errors $D_{\pi}(\hat{\boldsymbol{\theta}})$ and $D(\boldsymbol{f}_{\hat{\boldsymbol{\theta}}})$ become smaller as $\|\Delta\boldsymbol{\theta}\|$ decreases. This shows that if $\boldsymbol{w}$ and $b$ are optimally chosen, the rate of convergence of the TOP kernel is much faster than that of the plug-in estimate.

This result is closely related to large sample performances: Assuming that $\hat{\boldsymbol{\theta}}$ is a $n^{1/2}$ consistent estimator with asymptotic normality (e.g. the maximum likelihood estimator), we have $\|\Delta\boldsymbol{\theta}\| = O_p(n^{-1/2})$[7], where $O_p$ denotes stochastic order cf. [1]. So we can directly derive the convergence order as $D_{\pi}(\hat{\boldsymbol{\theta}}) = O_p(n^{-1/2})$ and $D(\boldsymbol{f}_{\hat{\boldsymbol{\theta}}}) = O_p(n^{-1})$. By using the relation (2.4), it follows that $R_{\pi}(\hat{\boldsymbol{\theta}}) - L^* = O_p(n^{-1/2})$ and $R(\boldsymbol{f}_{\hat{\boldsymbol{\theta}}}) - L^* = O_p(n^{-1})$.[5] Therefore, the TOP kernel has a much better convergence rate in $R(\boldsymbol{f}_{\hat{\boldsymbol{\theta}}})$, which is a strong motivation to use the TOP kernel instead of plug-in estimate.

However, we must notice that this fast rate is possible only when the optimal linear classifier is combined with the TOP kernel. Since non-optimal linear classifiers typically have the rate $O_p(n^{-1/2})$ [1], the overall rate is dominated by the slower rate and turns out to be $O_p(n^{-1/2})$. But this theoretical analysis is still meaningful, because it shows the existence of a very efficient linear boundary in the TOP feature space. This result encourages practical efforts to improve linear boundaries by engineering loss functions and regularization terms with e.g. cross validation, bootstrapping or other model selection criteria [1].

Exponential Family: A Special Case  When the distribution of two classes belong to the exponential family, the TOP kernel can achieve an even better result than shown above. Distributions of the exponential family can be written as $q(\boldsymbol{x}, \boldsymbol{\eta}) = \exp(\boldsymbol{\eta}^\top \boldsymbol{t}(\boldsymbol{x}) + \psi(\boldsymbol{\eta}))$, where $\boldsymbol{t}(\boldsymbol{x})$ is a vector-valued function called sufficient statistics and $\psi(\boldsymbol{\eta})$ is a normalization factor [4]. Let $\alpha$ denote the parameter for class prior probability of the positive model $P(y = +1)$. Then, the probabilistic model is described as

$$p(\boldsymbol{x}, y = +1|\boldsymbol{\theta}) = \alpha q_{+1}(\boldsymbol{x}, \boldsymbol{\eta}_{+1}), \;\; p(\boldsymbol{x}, y = -1|\boldsymbol{\theta}) = (1 - \alpha)q_{-1}(\boldsymbol{x}, \boldsymbol{\eta}_{-1}),$$

where $\boldsymbol{\theta} = \{\alpha, \boldsymbol{\eta}_{+1}, \boldsymbol{\eta}_{-1}\}$. The posterior log-odds reads

$$v(\boldsymbol{x}, \boldsymbol{\theta}) = \boldsymbol{\eta}_{+1}^\top \boldsymbol{t}_{+1}(\boldsymbol{x}) + \psi_{+1}(\boldsymbol{\eta}_{+1}) - \boldsymbol{\eta}_{-1}^\top \boldsymbol{t}_{-1}(\boldsymbol{x}) - \psi_{-1}(\boldsymbol{\eta}_{-1}) + \log \frac{\alpha}{1 - \alpha}. \quad (4.9)$$

The TOP feature vector is described as

$$f_{\hat{\boldsymbol{\theta}}}(\boldsymbol{x}) = (v(\boldsymbol{x}, \hat{\boldsymbol{\theta}}), \partial_\alpha v(\boldsymbol{x}, \hat{\boldsymbol{\theta}}), \nabla_{\boldsymbol{\eta}_{+1}} v(\boldsymbol{x}, \hat{\boldsymbol{\theta}})^\top, \nabla_{\boldsymbol{\eta}_{-1}} v(\boldsymbol{x}, \hat{\boldsymbol{\theta}})^\top)^\top.$$

where $\nabla_{\boldsymbol{\eta}_s} v(\boldsymbol{x}, \hat{\boldsymbol{\theta}}) = s(\boldsymbol{t}_s(\boldsymbol{x}) + \nabla_{\boldsymbol{\eta}_s} \psi_s(\hat{\boldsymbol{\eta}}_s))$ for $s = \{+1, -1\}$. So, when $\boldsymbol{w} = (1, 0, \boldsymbol{\eta}_{+1}^* - \hat{\boldsymbol{\eta}}_{+1}, \boldsymbol{\eta}_{-1}^* - \hat{\boldsymbol{\eta}}_{-1})^\top$ and $b$ is properly set, the true log-odds $F^{-1}(P(y = +1|\boldsymbol{x}, \boldsymbol{\theta}^*))$ can be constructed as a linear function in the feature space (4.1). Thus $D(\boldsymbol{f}_{\hat{\boldsymbol{\theta}}}) = 0$ and $R(\boldsymbol{f}_{\hat{\boldsymbol{\theta}}}) = L^*$. Furthermore, since each feature is represented as a linear function of sufficient statistics $\boldsymbol{t}_{+1}(\boldsymbol{x})$ and $\boldsymbol{t}_{-1}(\boldsymbol{x})$, one can construct an equivalent feature space as $(\boldsymbol{t}_{+1}(\boldsymbol{x})^\top, \boldsymbol{t}_{-1}(\boldsymbol{x})^\top)^\top$ without knowing $\hat{\boldsymbol{\theta}}$. This result is important because all graphical models without hidden states can be represented as members of the exponential family, for example markov models [4].

## 5  Experiments on Protein Data

In order to illustrate that the TOP kernel works well for real-world problems, we will show the results on protein classification. The protein sequence data is obtained from the Superfamily website.[6] This site provides sequence files with different degrees of redundancy filtering; we used the one with 10% redundancy filtering. Here, 4541 sequences are hierarchically labeled into 7 classes, 558 folds, 845 superfamilies and 1343 families according to the SCOP(1.53) scheme. In our experiment, we determine the top category "classes" as the learning target. The numbers of sequences in the classes are listed as 791, 1277, 1015, 915, 84, 76, 383. We only use the first 4 classes, and 6 two-class problems are generated from all pairs among the 4 classes. The 5th and 6th classes are not used because the number of examples is too small. Also, the 7th class is not used because this class is quite different from the others and too easy to classify. In each two-class problem, the examples are randomly divided into 25% training set, 25% validation set and 50% test set. The validation set is used for model selection.

As a probabilistic model for protein sequences, we make use of hidden markov models [2] with fully connected states.[7] The Baum-Welch algorithm (e.g. [2]) is used for maximum likelihood training. To construct FK and TOP kernels, the derivatives with respect to all parameters of the HMMs from both classes are included. The derivative with respect to the class prior probability is included as well: Let $q(\boldsymbol{x}, \boldsymbol{\xi})$ be the probability density function of a HMM. Then, the marginal distribution is written as $p(\boldsymbol{x}|\hat{\boldsymbol{\theta}}) = \hat{\alpha} q(\boldsymbol{x}, \hat{\boldsymbol{\xi}}_{+1}) + (1 - \hat{\alpha}) q(\boldsymbol{x}, \hat{\boldsymbol{\xi}}_{-1})$, where $\alpha$ is a parameter corresponding to the class prior. The feature vector of FK consists of the following:

$$\nabla_{\boldsymbol{\xi}_s} \log p(\boldsymbol{x}|\hat{\boldsymbol{\theta}}) = P(y = s|\boldsymbol{x}, \hat{\boldsymbol{\theta}}) \nabla_{\boldsymbol{\xi}_s} \log q(\boldsymbol{x}, \hat{\boldsymbol{\xi}}_s) \qquad s \in \{-1, +1\} \quad (5.1)$$

$$\partial_\alpha \log p(\boldsymbol{x}|\hat{\boldsymbol{\theta}}) = \frac{1}{\hat{\alpha}} P(y = +1|\boldsymbol{x}, \hat{\boldsymbol{\theta}}) - \frac{1}{1 - \hat{\alpha}} P(y = -1|\boldsymbol{x}, \hat{\boldsymbol{\theta}}), \quad (5.2)$$

while the feature vector of TOP includes $\nabla_{\boldsymbol{\xi}_s} v(\boldsymbol{x}, \hat{\boldsymbol{\theta}}) = s \nabla_{\boldsymbol{\xi}_s} \log q(\boldsymbol{x}, \hat{\boldsymbol{\xi}}_s)$ $s = \{+1, -1\}$.[8] We obtained $\hat{\boldsymbol{\xi}}_{+1}$ and $\hat{\boldsymbol{\xi}}_{-1}$ from the training examples of respective classes and set $\hat{\alpha} = 0.5$. In the definition of the TOP kernel (4.5), we did not include any normalization of feature vectors. However, in practical situations, it is effective to normalize features for improving classification performance. Here, each feature of the TOP kernel is normalized to have mean 0 and variance 1. Also in FK, we normalized the features in the same way instead of using the Fisher information matrix, because it is difficult to estimate it reliably in a high dimensional parameter space.Both, the TOP kernel and FK are combined with SVMs with bias terms.

When classifying with HMMs, one observes the difference of the log-likelihoods for the two classes and discriminates by thresholding at an appropriate value. Theoretically, this threshold should be determined by the (true) class prior probability. But, this is typically not available. Furthermore the estimation of the prior probability from training data often leads to poor results [2]. To avoid this problem, the threshold is determined such that the false positive rate and the false negative rate are equal in the test set. This threshold is determined in the same way for FK-SVMs and TOP-SVMs.

The hybrid HMM-TOP-SVM system has several model parameters: the number of HMM states, the pseudo count value [2] and the regularization parameter $C$ of the SVM. We determine these parameters as follows: First, the number of states and the pseudo count value are determined such that the error of the HMM on the validation set (i.e. validation error) is minimized. Based on the chosen HMM model, the parameter $C$ is determined such that the validation error of TOP-SVM is minimized. Here, the number of states and the pseudo count value are chosen from $\{3, 5, 7, 10, 15, 20, 30, 40, 60\}$ and $\{10^{-10}, 10^{-7}, 10^{-5}, 10^{-4}, 10^{-3}, 10^{-2}\}$, respectively. For $C$, 15 equally spaced points on the log scale are taken from $[10^{-4}, 10^1]$. Note that the model selection is performed in the same manner for the Fisher kernel as well.

The error rates over 15 different training/validation/test divisions are shown in Figure 1 and 2. The results of statistical tests are shown in Table 1 as well. Compared with the plug-in estimate, the Fisher kernel performed significantly better in several settings (i.e. 1-3, 2-3, 3-4). This result partially agrees with observations in [6]. However, our TOP approach significantly outperforms the Fisher kernel: According to the Wilcoxson signed ranks test, the TOP kernel was significantly better

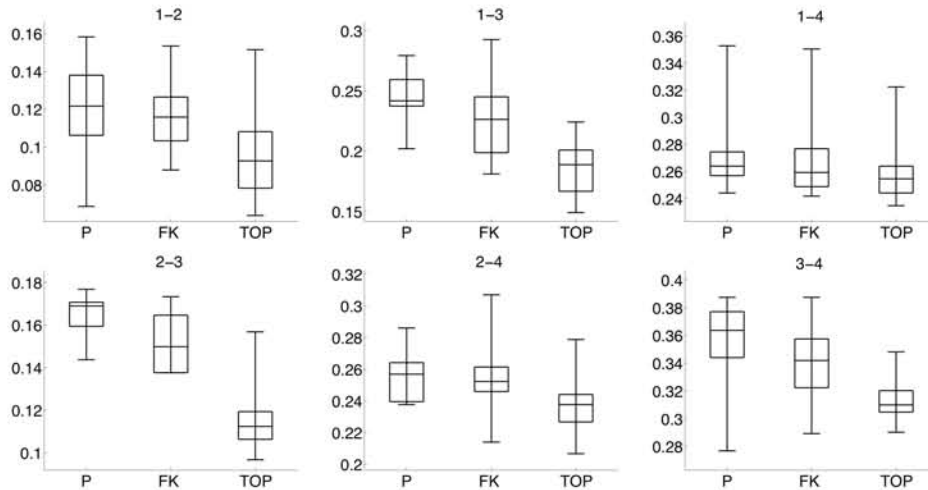

Figure 1: The error rates of SVMs with two feature extractors in the protein classification experiments. The labels 'P','FK','TOP' denote the plug-in estimate, the Fisher kernel and the TOP kernel, respectively. The title on each subfigure shows the two protein classes used for the experiment.

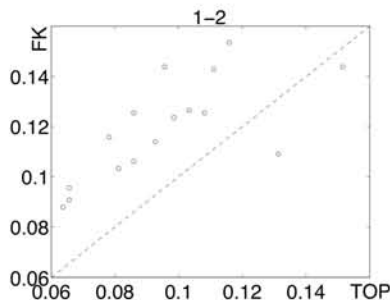

Figure 2: Comparison of the error rates of the Fisher kernel and the TOP kernel in discrimination between class 1 and 2. Every point corresponds to one of 15 different training/validation/test set splits. Except two cases, the TOP kernel achieves smaller error rates.

in all settings. Also, the t-test judged that the difference is significant except for 1-4 and 2-4. This indicates that the TOP kernel was able to capture discriminative information better than the Fisher kernel.

## 6  Conclusion

In this study, we presented the new discriminative TOP kernel derived from probabilistic models. Since the theoretical framework for such kernels has so far not been established, we proposed two performance measures to analyze them and gave bounds and rates to gain a better insight into model dependent feature extractors from probabilistic models. Experimentally, we showed that the TOP kernel compares favorably to FK in a realistic protein classification experiment. Note that Smith and Gales[8] have shown that a similar approach works excellently in speech recognition tasks as well. Future research will focus on constructing small sample bounds for the TOP kernel to extend the validity of this work. Since other nonlinear transformations $F$ are possible for obtaining different and possibly better features, we will furthermore consider to learn the nonlinear transformation $F$ from training samples. An interesting point is that so far TOP kernels perform local linear approximations, it would be interesting to move in the direction of local or even

Table 1: P-values of statistical tests in the protein classification experiments. Two kinds of tests, t-test (denoted as T in the table) and Wilcoxson signed ranks test (i.e. WX), are used. When the difference is significant (p-value $< 0.05$), a single star $*$ is put beside the value. Double stars $**$ indicate that the difference is very significant (p-value $< 0.01$).

| Methods | Test | 1-2 | 1-3 | 1-4 |
|---------|------|-----|-----|-----|
| P, FK | T | 0.95 | 0.14 | 0.78 |
|       | WX | 0.85 | 0.041* | 0.24 |
| P, TOP | T | 0.015* | $1.7 \times 10^{-8**}$ | 0.11 |
|        | WX | $4.3 \times 10^{-4**}$ | $6.1 \times 10^{-5**}$ | 0.030* |
| FK, TOP | T | 0.0093** | $2.2 \times 10^{-4**}$ | 0.21 |
|         | WX | $8.5 \times 10^{-4**}$ | $6.1 \times 10^{-5**}$ | 0.048* |

| Methods | Test | 2-3 | 2-4 | 3-4 |
|---------|------|-----|-----|-----|
| P, FK | T | 0.0032** | 0.79 | 0.12 |
|       | WX | 0.0040** | 0.80 | 0.026* |
| P, TOP | T | $3.0 \times 10^{-12**}$ | 0.059 | $5.3 \times 10^{-5**}$ |
|        | WX | $6.1 \times 10^{-5**}$ | 0.035* | $3.1 \times 10^{-4**}$ |
| FK, TOP | T | $2.6 \times 10^{-8**}$ | 0.079 | 0.0031** |
|         | WX | $6.1 \times 10^{-5**}$ | 0.0034** | $1.8 \times 10^{-4**}$ |

global nonlinear expansions.

Acknowledgments   We thank T. Tanaka, M. Sugiyama, S.-I. Amari, K. Karplus, R. Karchin, F. Sohler and A. Zien for valuable discussions. Moreover, we gratefully acknowledge partial support from DFG (JA 379/9-1, MU 987/1-1) and travel grants from EU (Neurocolt II).

## Footnotes

[1] In classification by plug-in estimate, $\boldsymbol{x}$ is classified by thresholding the posterior probability $\hat{y} = \text{sign}(P(y = +1|\boldsymbol{x}, \hat{\boldsymbol{\theta}}) - 0.5)$ [1].

[2] In practice, some variants of the Fisher kernel are used. For example, if the derivative of each class distribution, not marginal, is taken, the feature vector of FK is quite similar to that of our kernel. However, these variants should be deliberately discriminated from the Fisher kernel in theoretical discussions. Throughout this paper including experiments, we adopt the original definition of the Fisher kernel from [6].

[3] Notice that $F^{-1}(t) = \log t - \log(1 - t)$

[4] One can easily derive TOP kernels from higher order Taylor expansions. However, we will only deal with the first order expansion here, because higher order expansions would induce extremely high dimensional feature vectors in practical cases.

[5] For detailed discussion about the convergence orders of classification error, see Chapter 6 of [1]

[6] http://stash.mrc-lmb.cam.ac.uk/SUPERFAMILY/

[7]Several HMM models have been engineered for protein classification [2]. However, we do not use such HMMs because the main purpose of experiment is to compare FK and TOP.

[8]$\partial_\alpha v(\boldsymbol{x}, \hat{\boldsymbol{\theta}})$ is a constant which does not depend on $\boldsymbol{x}$. So it is not included in the feature vector.

# References

[1] L. Devroye, L. Györfi, and G. Lugosi. A Probabilistic Theory of Pattern Recognition. Springer, 1996.

[2] R. Durbin, S. Eddy, A. Krogh, and G. Mitchison. Biological Sequence Analysis: Probabilistic Models of Proteins and Nucleic Acids. Cambridge University Press, 1998.

[3] K. Fukunaga. Introduction to Statistical Pattern Recognition. Academic Press, San Diego, 2nd edition, 1990.

[4] D. Geiger and C. Meek. Graphical models and exponential families. Technical Report MSR-TR-98-10, Microsoft Research, 1998.

[5] T.S. Jaakkola, M. Diekhans, and D. Haussler. A discriminative framework for detecting remote protein homologies. J. Comp. Biol., 7:95–114, 2000.

[6] T.S. Jaakkola and D. Haussler. Exploiting generative models in discriminative classifiers. In M.S. Kearns, S.A. Solla, and D.A. Cohn, editors, Advances in Neural Information Processing Systems 11, pages 487–493. MIT Press, 1999.

[7] N. Murata, S. Yoshizawa, and S. Amari. Network information criterion — determining the number of hidden units for an artificial neural network model. IEEE Trans. Neural Networks, 5:865–872, 1994.

[8] N. Smith and M. Gales. Speech recognition using SVMs. In T.G. Dietterich, S. Becker, and Z. Ghahramani, editors, Advances in Neural Information Processing Systems 14. MIT Press, 2002. to appear.
